# Submodular Multi-Label Learning

**James Petterson**
NICTA/ANU
Canberra, Australia

**Tiberio Caetano**
NICTA/ANU
Sydney/Canberra, Australia

## Abstract

In this paper we present an algorithm to learn a multi-label classifier which attempts at directly optimising the $F$-score. The key novelty of our formulation is that we explicitly allow for assortative (submodular) pairwise label interactions, i.e., we can leverage the co-ocurrence of pairs of labels in order to improve the quality of prediction. Prediction in this model consists of minimising a particular submodular set function, what can be accomplished exactly and efficiently via graph-cuts. Learning however is substantially more involved and requires the solution of an intractable combinatorial optimisation problem. We present an approximate algorithm for this problem and prove that it is sound in the sense that it never predicts incorrect labels. We also present a nontrivial test of a sufficient condition for our algorithm to have found an optimal solution. We present experiments on benchmark multi-label datasets, which attest the value of the proposed technique. We also make available source code that enables the reproduction of our experiments.

## 1  Introduction

Research in multi-label classification has seen a substantial growth in recent years (e.g., [1, 2, 3, 4]). This is due to a number of reasons, including the increase in availability of multi-modal datasets and the emergence of crowdsourcing, which naturally create settings where multiple interpretations of a given input observation are possible (multiple labels for a single instance). Also many classical problems are inherently multi-label, such as the categorisation of documents [5], gene function prediction [6] and image tagging [7].

There are two desirable aspects in a multi-label classification system. The first is that a prediction should ideally be good both in terms of precision and recall: we care not only about predicting as many of the correct labels as possible, but also as few non-correct labels as possible. One of the most popular measures for assessing performance is therefore the $F$-score, which is the harmonic mean of precision and recall [8]. The second property we wish is that, both during training and also at test time, the algorithm should ideally take into account possible dependencies between the labels. For example, in automatic image tagging, if labels *ocean* and *ship* have high co-occurrence frequency in the training set, the model learned should somehow boost the chances of predicting *ocean* if there is strong visual evidence for the label *ship* [9].

In this paper we present a method that directly addresses these two aspects. First, we explicitly model the dependencies between pairs of labels, albeit restricting them to be submodular (in rough terms, we model only the positive pairwise label correlations). This enables exact and efficient prediction at test time, since finding an optimal subset of labels reduces to the minimisation of a particular kind of submodular set function which can be done efficiently via graph-cuts. Second, our method directly attempts at optimising a convex surrogate of the $F$-score. This is because we draw on the max-margin structured prediction framework from [10], which, as we will see, enables us to optimise a convex upper bound on the loss induced by the $F$-score. The critical technical contribution of the paper is a constraint generation algorithm for loss-augmented inference where the scoring of the pair (input-output) is a submodular set function and the loss is derived from the $F$-score. This

is what enables us to fit our model into the estimator from [10]. Our constraint generation algorithm is only approximate since the problem is intractable. However we give theoretical arguments supporting our empirical findings that the algorithm is not only very accurate in practice, but in the majority of our real-world experiments it actually produces a solution which is exactly optimal. We compare the proposed method with other benchmark methods on publicly available multi-label datasets, and results favour our approach. We also provide source code that enables the reproduction of all the experiments presented in this paper.

**Related Work.** A convex relaxation for $F$-measure optimisation in the multi-label setting was proposed recently in [11]. This can be seen as a particular case of our method when there are no explicit label dependencies. In [12] the authors propose quite general tree and DAG-based dependencies among the labels and adapt decoding algorithms from signal processing to the problem of finding predictions consistent with the structures learned. In [13] graphical models are used to impose structure in the label dependencies. Both [12] and [13] are in a sense complementary to our method since we do not enforce any particular graph topology on the labels but instead we limit the nature of the interactions to be submodular. In [14] the authors study the multi-label problem under the assumption that prior knowledge on the density of label correlations is available. They also use a max-margin framework, similar in spirit to our formulation. A quite simple and basic strategy for multi-label problems is to treat them as multiclass classification, effectively ignoring the relationships between the labels. One example in this class is the Binary Method [15]. The RAkEL algorithm [16] uses instead an ensemble of classifiers, each learned on a random subset of the label set. In [17] the authors propose a Bayesian CCA model and apply it to multi-label problems by enforcing group sparsity regularisation in order to capture information about label co-occurrences.

## 2    The Model

Let $x \in \mathcal{X}$ be a vector of dimensionality $D$ with the features of an instance (say, an image); let $y \in \mathcal{Y}$ be a *set of labels* for an instance (say, tags for an image), from a fixed dictionary of $V$ possible labels, encoded as $y \in \{0,1\}^V$. For example, $y = [1\ 1\ 0\ 0]$ denotes the first and second labels of a set of four. We assume we are given a training set $\{(x^n, y^n)\}_{n=1}^N$, and our task is to estimate a map $f : \mathcal{X} \to \mathcal{Y}$ that has good agreement with the training set but also generalises well to new data. In this section we define the class of functions $f$ that we will consider. In the next section we define the learning algorithm, i.e., a procedure to find a specific $f$ in the class.

### 2.1    The Loss Function Derived from the $F$-Score

Our notion of 'agreement' with the training set is given by a loss function. We focus on maximising the average over all instances of $F$, a score that considers both *precision* and *recall* and can be written in our notation as

$$F = \frac{1}{N} \sum_{n=1}^{N} \frac{2\, p(y^n, \bar{y}^n) r(y^n, \bar{y}^n)}{p(y^n, \bar{y}^n) + r(y^n, \bar{y}^n)}, \quad \text{where} \ \ p(y, \bar{y}) = \frac{|y \odot \bar{y}|}{|\bar{y}|} \ \ \text{and} \ \ r(y, \bar{y}) = \frac{|y \odot \bar{y}|}{|y|}$$

Here $\bar{y}^n$ denotes our prediction for input instance $n$, $y^n$ is the corresponding ground-truth, $\odot$ denotes the element-wise product and $|u|$ denotes the 1-norm of vector $u$ (in our case the number of 1s since $u$ will always be binary). Since our goal is to *maximise* the $F$-score a suitable choice of loss function is $\Delta(y, \bar{y}) = 1 - F(y, \bar{y})$, which is the one we adopt in this paper. The loss for a single prediction is therefore

$$\Delta(y, \bar{y}) = 1 - 2\,|y \odot \bar{y}|/(|y| + |\bar{y}|) \tag{1}$$

### 2.2    Feature Maps and Parameterisation

We assume that the prediction for a given input $x$ is a maximiser of a score that encodes both the unary dependency between labels and instances as well as the pairwise dependencies between labels:

$$\bar{y} \in \operatorname*{argmax}_{y \in \mathcal{Y}} y^T A y \tag{2}$$

where $A$ is an upper-triangular matrix scoring the pair $(x, y)$, with diagonal elements $A_{ii} = \langle x, \theta_i^1 \rangle$, where $x$ is the input feature vector and $\theta_i^1$ is a parameter vector that defines how label $i$ weighs each feature of $x$. The off-diagonal elements are $A_{ij} = C_{ij}\theta_{ij}$, where $C_{ij}$ is the normalised counts of co-occurrence of labels $i$ and $j$ in the training set, and $\theta_{ij}^2$ the corresponding scalar parameter *which is forced to be non-negative*. This will ensure that the off-diagonal entries of $A$ are non-negative and therefore that problem 2 consists of the maximisation of a supermodular function (or, equivalently, the minimisation of a submodular function), which can be solved efficiently via graph-cuts. We also define the complete parameter vectors $\theta^1 := [\ldots \theta_i^{1^T} \ldots]^T$, $\theta^2 := [\ldots \theta_{ij}^2 \ldots]^T$ and $\theta = [\theta^{1^T}\ \theta^{2^T}]^T$, as well as the complete feature maps $\phi_1(x, y) = vec(x \otimes y)$, $\phi_2(y) = vec(y \otimes y)$ and $\phi(x, y) = [\phi_1^T(x, y)\ \phi_2^T(y)]^T$. This way the score in expression 2 can be written as $y^T A y = \langle \phi(x, y), \theta \rangle$. Note that the dimensionality of $\theta^2$ is the number of non-zero elements of matrix $C$–in this setting that is $\binom{V}{2}$, but it can be reduced by setting to zero elements of $C$ below a specified threshold.

## 3 Learning Algorithm

**Optimisation Problem.** Direct optimisation of the loss defined in equation 1 is a highly intractable problem since it is a discrete quantity and our parameter space is continuous. Here we will follow the program in [10] and instead construct a convex upper bound on the loss function, which can then be attacked using convex optimisation tools. The purpose of learning will be to solve the following convex optimisation problem

$$[\theta^*, \xi^*] = \operatorname*{argmin}_{\theta, \xi} \left[ \frac{1}{N} \sum_{n=1}^N \xi_n + \frac{\lambda}{2} \|\theta\|^2 \right] \tag{3a}$$

$$\text{s.t. } \langle \phi(x^n, y^n), \theta \rangle - \langle \phi(x^n, y), \theta \rangle \geq \Delta(y, y^n) - \xi_n, \tag{3b}$$
$$\xi_n \geq 0, \forall n, y \neq y^n.$$

This is the margin-rescaling estimator for structured support vector machines [10]. The constraints immediately imply that the optimal solution will be such that $\xi_n^* \geq \Delta(\operatorname{argmax}_y \langle \phi(x^n, y), \theta^* \rangle, y^n)$, and therefore the minimum value of the objective function upper bounds the loss, thus motivating the formulation. Since there are exponentially many constraints, we follow [10] in adopting a constraint generation strategy, which starts by solving the problem with no constraints and iteratively adding the most violated constraint for the current solution of the optimisation problem. This is guaranteed to find an $\epsilon$-close approximation of the solution of (3) after including only a polynomial ($O(\epsilon^{-2})$) number of constraints [10]. At each iteration we need to maximise the violation margin $\xi_n$, which from the constraints 3b reduces to

$$y_n^* \in \operatorname*{argmax}_{y \in \mathcal{Y}} \left[ \Delta(y, y^n) + \langle \phi(x^n, y), \theta \rangle \right] \tag{4}$$

**Learning Algorithm.** The learning algorithm is described in Algorithm 1 (requires as subroutine Algorithm 2). Algorithm 1 describes a particular convex solver based on bundle methods (BMRM [18]), which we use here. Other solvers could have been used instead. Our contribution lies not here, but in the routine of constraint generation for Algorithm 1, which is described in Algorithm 2.

BMRM requires the solution of constraint generation and the value of the objective function for the slack corresponding to the constraint generated, as well as its gradient. Soon we will discuss constraint generation. The other two ingredients we describe here. The slack at the optimal solution is

$$\xi_n^* = \Delta(y_n^*, y^n) + \langle \phi(x^n, y_n^*), \theta \rangle - \langle \phi(x^n, y^n), \theta \rangle \tag{5}$$

thus the objective function from (3) becomes

$$\frac{1}{N} \sum_n \Delta(y_n^*, y^n) + \langle \phi(x^n, y_n^*), \theta \rangle - \langle \phi(x^n, y^n), \theta \rangle + \frac{\lambda}{2} \|\theta\|^2, \tag{6}$$

whose gradient is

$$\lambda \theta - \frac{1}{N} \sum_n (\phi(x^n, y^n) - \phi(x^n, y_n^*)) \tag{7}$$

**Algorithm 1** Bundle Method for Regularised Risk Minimisation (BMRM)

1: **Input:** training set $\{(x^n, y^n)\}_{n=1}^N$, $\lambda$, **Output:** $\theta$
2: Initialize $i = 1$, $\theta_1 = 0$, max$= -\infty$
3: **repeat**
4:    **for** $n = 1$ **to** $N$ **do**
5:       Compute $y_n^*$ ($y_{k_{max}}^{*n}$ returned by Algorithm 2.)
6:    **end for**
7:    Compute gradient $g_i$ (equation (7)) and objective $o_i$ (equation (6))
8:    $\theta_{i+1} := \text{argmin}_\theta \frac{\lambda}{2} \|\theta\|^2 + \max(0, \max_{j \leq i} \langle g_j, \theta \rangle + o_j)$; $i \leftarrow i + 1$
9: **until** converged (see [17])
10: **return** $\theta$

**Algorithm 2** Constraint Generation

1: **Input:** $(x^n, y^n)$, $\theta$, $V$, **Output:** $y_{k_{max}}^{*n}$
2: $k = 0$
3: $A_{ij}^{[k],n} = \langle \theta_{ij}, C_{ij} \rangle$ (for all $i, j : i \neq j$)
4: **while** $k \leq V$ **do**
5:    $diag(A^{[k],n}) = diag(A) - \frac{2y^n}{k + \|y^n\|^2}$
6:    $y_k^{*n} = \text{argmax}_y \, y^T A^{[k],n} y$ (graph-cuts)
7:    **if** $|y_k^{*n}| > k$ **then**
8:       $k_{max} = |y_k^{*n}|$; $k = k_{max}$
9:    **else if** $|y_k^{*n}| = k$ **then**
10:       $k_{max} = |y_k^{*n}|$; $k = k_{max} + 1$
11:    **else**
12:       $k = k + 1$
13:    **end if**
14: **end while**
15: **return** $y_{k_{max}}^{*n}$

Expressions (6) and (7) are then used in Algorithm 1.

**Constraint Generation.** The most challenging step consists of solving the constraint generation problem. Constraint generation for a given training instance $n$ consists of solving the combinatorial optimisation problem in expression 4, which, using the loss in equation 1, as well as the correspondence $y^T A y = \langle \phi(x, y), \theta \rangle$, can be written as

$$y^{*n} \in \underset{y}{\text{argmax}} \, y^T A^n(y) y \tag{8}$$

where $diag(A^n) = diag(A) - \frac{2 \, y^n}{|y| + |y^n|}$ and offdiag$(A^n) = $ offdiag$(A)$. Note that the matrix $A^n$ depends on $y$. More precisely, a subset of its diagonal elements (those $A_{ii}^n$ for which $y^n(i) = 1$) depends on the quantity $|y|$, i.e., the number of nonzero elements in $y$. This makes solving problem 8 a formidable task. If $A^n$ were independent of $y$, then eq. 8 could be solved exactly and efficiently via graph-cuts, just as our prediction problem in equation 2. A naïve strategy would be to aim for solving problem 8 $V$ times, one for each value of $|y|$, and constraining the optimisation to only include elements $y$ such that $|y|$ is fixed. In other words, we can partition the optimisation problem into $k$ optimisation problems conditioned on the sets $\mathcal{Y}_k := \{y : |y| = k\}$:

$$\max_y y^T A(y) y = \max_k \max_{y \in \mathcal{Y}_k} y^T A^{[k],n} y \tag{9}$$

where $A^{[k],n}$ denotes the particular matrix $A^n$ that we obtain when $|y| = k$. However the inner maximization above, i.e., the problem of maximising a supermodular function (or minimising a submodular function) subject to a cardinality constraint, is itself NP-hard [19]. We therefore do not follow this strategy, but instead seek a polynomial-time algorithm that in practice will give us an optimal solution most of the time.

Algorithm 2 describes our algorithm. In the worst case it calls graph-cuts $O(V)$ times, so the total complexity is $O(V^4)$.[1] The algorithm essentially searches for the largest $k$ such that solving $\text{argmax}_y y^T A^{[k],n} y$ returns a solution with $k$ 1s. We call the $k$ obtained $k_{max}$, and the corresponding solution $y_{k_{max}}^{*n}$. Observe the fact that, as $k$ increases during the execution of the algorithm, $A_{ii}^n$ increases for those $i$ where $y^n(i) = 1$. The increment observed when $k$ increases to $k'$ is

$$\epsilon_k^{k'} := A_{ii}^{[k'],n} - A_{ii}^{[k],n} = 2 \frac{k' - k}{(k' + |y^n|)(k + |y^n|)} \tag{10}$$

which is always a positive quantity. Although this algorithm is not provably optimal, Theorem 1 guarantees that it is sound in the sense that it never predicts incorrect labels. In the

next section we present additional evidence supporting this algorithm, in the form of a test that if positive guarantees the solution obtained is optimal.

We call a solution $y'$ a *partially optimal* solution of $\text{argmax}_y \, y^T A^n(y)y$ if the labels it predicts as being present are indeed present in an optimal solution, i.e., if for those $i$ for which $y'(i) = 1$ we also have $y^{*n}(i) = 1$, for some $y^{*n} \in \text{argmax}_y \, y^T A^n(y)y$. Equivalently, we can write $y' \odot y^{*n} = y'$. We have the following result

**Theorem 1** *Upon completion of Algorithm 2, $y_{k_{max}}^{*n}$ is a partially optimal solution of* $\text{argmax}_y \, y^T A^n(y)y$.

The proof is in the Appendix A. The theorem means that whenever the algorithm predicts the presence of a label, it does so correctly; however there may be labels not predicted which are in fact present in the corresponding optimal solution.

## 4   Certificate of Optimality

As empirically verified in our experiments in section 5, our constraint generation algorithm (Algorithm 2) is indeed quite accurate: most of the time the solution obtained is optimal. In this section we present a test that if positive guarantees that an optimal solution has been obtained (i.e., a certificate of optimality). This can be used to generate empirical lower bounds on the probability that the algorithm returns an optimal solution (we explore this possibility in the experimental section).

We start by formalising the situation in which the algorithm will fail. Let $Z := \{i : y_{k_{max}}^{*n}(i) = 0\}$, and $\mathcal{P}_Z$ be the power set of $Z$ ($Z$ for 'zeros'). Let $O := \{i : y_{k_{max}}^{*n}(i) = 1\}$ ($O$ for 'ones'). Then the algorithm will fail if there exists $\alpha \in \mathcal{P}_Z$ such that

$$\underbrace{\sum_{i,j \in \alpha; i \neq j} A_{ij}^n}_{(a)} + \underbrace{\sum_{i \in \alpha, j \in O} A_{ij}}_{(b)} + \underbrace{\sum_{i \in \alpha} A_{ii}^{[k_{max}+|\alpha|],n}}_{(c)} + \underbrace{\epsilon_{k_{max}}^{k_{max}+|\alpha|} |y^n \odot y_{k_{max}}^{*n}|}_{(d)} > 0 \qquad (11)$$

The above expression describes the situation in which, starting with $y_{k_{max}}^{*n}$, if we insert $|\alpha|$ $1s$ in the indices defined by index set $\alpha$, we will obtain a new vector $y'$ which is a feasible solution of $\text{argmax}_y \, y^T A^n(y)y$ and yet has strictly larger score than solution $y_{k_{max}}^{*n}$. This can be understood by looking closely into each of the sums in expression 11. Sums $(a)$ and $(b)$ describe the increase in the objective function due to the inclusion of off-diagonal terms. Both $(a)$ and $(b)$ are non-negative due to the submodularity assumption. Term $(c)$ is the sum of the diagonal terms corresponding to the newly introduced $1s$ of $y'$. Term $(c)$ is negative or zero, since each term in the sum is negative or zero (otherwise $y_{k_{max}}^{*n}$ would have included it). Finally, term $(d)$ is non-negative, being the total increase in the diagonal elements of $O$ due to the inclusion of $|\alpha|$ additional $1s$. We can write $(c)$ as

$$\underbrace{\sum_{i \in \alpha} A_{ii}^{[k_{max}+|\alpha|],n}}_{(c)} = \underbrace{\sum_{i \in \alpha} A_{ii}^{[k_{max}],n}}_{(e)} + \underbrace{\sum_{i \in \alpha} (A_{ii}^{[k_{max}+|\alpha|],n} - A_{ii}^{[k_{max}],n})}_{(f)} \qquad (12)$$

and the last term can be bounded as

$$\sum_{i \in \alpha} (A_{ii}^{[k_{max}+|\alpha|],n} - A_{ii}^{[k_{max}],n}) \leq \underbrace{\epsilon_{k_{max}}^{k_{max}+|\alpha|} v_\alpha}_{(g)} \qquad (13)$$

where $v_\alpha = \min[|y^n| - |y^n \odot y_{k_{max}}^{*n}|, |\alpha|]$ is an upper bound on the number of indices $i \in \alpha$ such that $y^n(i) = 1$, and $\epsilon_{k_{max}}^{k_{max}+|\alpha|}$ is the increment in a diagonal element $i$ for which $y^n(i) = 1$ arising from increasing the cardinality of the solution from $k_{max}$ to $k_{max}+|\alpha|$. Incorporating bound 13 into equation 12, we get that $(c) \leq (e)+(g)$. We can then replace $(c)$ in inequality 11 by $(e)+(g)$, obtaining

$$\underbrace{\sum_{i,j \in \alpha; i \neq j} A_{ij}^n + \sum_{i \in \alpha, j \in O} A_{ij} + \sum_{i \in \alpha} A_{ii}^{[k_{max}],n}}_{:=\beta_{A,\alpha}} + \underbrace{\epsilon_{k_{max}}^{k_{max}+|\alpha|} v_\alpha + \epsilon_{k_{max}}^{k_{max}+|\alpha|} |y^n \odot y_{k_{max}}^{*n}|}_{:=\gamma_\alpha} > 0 \qquad (14)$$

**Algorithm 3** Compute $\max_\alpha \beta_{A,\alpha}$

1: **Input:** $A^{[k_{max}],n}$, $y^{*n}_{k_{max}}$, $V$,
2: **Output:** max
3: max $= -\infty$
4: $Z = \{i : y^{*n}_{k_{max}}(i) = 0\}$
5: $O = \{i : y^{*n}_{k_{max}}(i) = 1\}$
6: **for** $i \in Z$ **do**
7: $\quad O' = O \cup i$
8: $\quad$ rmax $= \max_{y:y_{O'}=1} y^T A^{[k_{max}],n} y$ (graph-cuts)
9: $\quad$ **if** rmax $>$ max **then**
10: $\quad\quad$ max $=$ rmax
11: $\quad$ **end if**
12: **end for**
13: max $=$ max $- \max_y y^T A^{[k_{max}],n} y$
14: **return** max

Table 1: Datasets. #train/#test denotes the number of observations used for training and testing respectively; $V$ is the number of labels and $D$ the dimensionality of the features; Avg is the average number of labels per instance.

| dataset | yeast | enron |
|---|---|---|
| **domain** | biology | text |
| **#train** | 1500 | 1123 |
| **#test** | 917 | 579 |
| **V** | 14 | 53 |
| **D** | 103 | 1001 |
| **Avg** | 4.23 | 3.37 |

We know that, regardless of $A$ or $\alpha$, $\beta_{A,\alpha} \leq 0$ (otherwise $y^{*n}_{k_{max}} \notin \text{argmax}_y y^T A^{[k_{max}],n} y$, since $\beta_{A,\alpha}$ is the increment in the objective function $y^T A^{[k_{max}],n} y$ obtained by adding $1s$ in the entries of $\alpha$). The key fact coming to our aid is that $\gamma_\alpha$ is 'small', and a weak upper bound is 2. This is because

$$\epsilon^{k_{max}+|\alpha|}_{k_{max}} v_\alpha + \epsilon^{k_{max}+|\alpha|}_{k_{max}} |y^n \odot y^{*n}_{k_{max}}| \leq \epsilon^{k_{max}+|\alpha|}_{k_{max}} |y^n| \leq \epsilon^V_{k_{max}} |y^n| \leq \epsilon^V_0 |y^n| =$$
$$= 2V|y^n|/((V+|y^n|)|y^n|) \leq 2 \tag{15}$$

(Note that if $|y^n| = 0$ then $\gamma_\alpha = 0$ and our algorithm will always return an optimal solution since $\beta_{A,\alpha} \leq 0$). Now, since $\beta_{A,\alpha} \leq 0$ for any $A$ and $\alpha \in \mathcal{P}_Z$, it suffices that we study the quantity $\max_\alpha \beta_{A,\alpha}$: if $\max_\alpha \beta_{A,\alpha} < -2$, then $\beta_{A,\alpha} < -2$ for any $\alpha \in \mathcal{P}_Z$. It is however very hard to understand theoretically the behaviour of the random variable $\max_\alpha \beta_{A,\alpha}$ even for a simplistic uniform i.i.d. assumption on the entries of $A$. This is because the domain of $\alpha$, $\mathcal{P}_Z$, is itself a random quantity that depends on the particular $A$ chosen. This makes computing even the expected value of $\max_\alpha \beta_{A,\alpha}$ an intractable task, let alone obtaining concentration of measure results that could give us upper bounds on the probability of condition 14 holding under the assumed distribution on $A$.

However, for a *given A* we can actually compute $\max_\alpha \beta_{A,\alpha}$ efficiently. This can be done with Algorithm 3. The algorithm effectively computes the gap between the scores of the optimal solution $y^{*n}_{k_{max}}$ and the highest scoring solution if one sets to 1 at least one of the zero entries in $y^{*n}_{k_{max}}$. It does so by solving graph-cuts constraining the solution $y$ to include the 1s present in $y^{*n}_{k_{max}}$ but additionally fixing one of the zero entries of $y^{*n}_{k_{max}}$ to 1 (lines 7-8). This is done for every possible zero entry of $y^{*n}_{k_{max}}$, and the maximum score is recorded (lines 7-11). The gap between this and the score of the optimal solution $y^{*n}_{k_{max}}$ is then returned (line 13). This will involve $V - k_{max}$ calls to graph-cuts, and therefore the total computational complexity is $O(V^4)$. Once we compute $\max_\alpha \beta_{A,\alpha}$, we simply test wether $\max_\alpha \beta_{A,\alpha} + \epsilon^{|V|}_{k_{max}} |y^n| > 0$ holds (we use $\epsilon^{|V|}_{k_{max}} |y^n|$ rather than 2 as an upper bound for $\gamma_\alpha$ because, as seen from (15), it is the tightest upper bound which still does not depend on $\alpha$ and therefore can be computed). We have the following theorem (proven in Appendix A)

**Theorem 2** *Upon completion of Algorithm 3, if* $\max_\alpha \beta_{A,\alpha} + \epsilon^{|V|}_{k_{max}} |y^n| \leq 0$*, then* $y^{*n}_{k_{max}}$ *is an optimal solution of* $\text{argmax}_y y^T A^n(y) y$*.*

## 5 Experimental Results

To evaluate our multi-label learning method we applied it to real-world datasets and compared it to state-of-the art methods.

**Datasets.** For the sake of reproducibility we focused in publicly available datasets, and to ensure that the label dependencies have a reasonable impact in the results we restricted the experiments to datasets with a sufficiently large average number of labels per instance. We

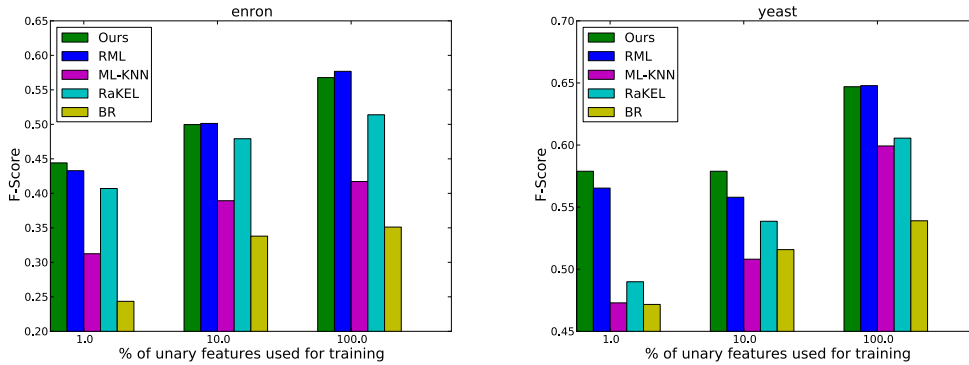

Figure 1: *F*-Score results on *enron* (left) and *yeast* (right), for different amounts of unary features. The horizontal axis denotes the proportion of the features used in training.

chose therefore two multilabel datasets from mulan:[2] *yeast* and *enron*. Table 1 describes them in more detail.

**Experimental setting.** The datasets used have very informative unary features, so to better visualise the contribution of the label dependencies to the model we trained using varying amounts (1%, 10% and 100%) of the original unary features. We compared our proposed method to RML[11] without reversion[3], which is essentially our model without the quadratic term, and to other state-of-the-art methods for which source code is publicly available – BR [15], RAkEL[16] and MLKNN[20].

**Model selection.** Our model has two parameters: $\lambda$, the trade-off between data fitting and good generalisation, and $c$, a scalar that multiplies $C$ to control the trade-off between the linear and the quadratic terms. For each experiment we selected them with 5-fold cross-validation on the training data. We also control the sparsity of $C$ by setting $C_{ij}$ to zero for all except the top most frequent pairs – this way we can reduce the dimensionality of $\theta^2$, avoiding an excessive number of parameters for datasets with large values of $V$. In our experiments we used 50% of the pairs with *yeast* and 5% with *enron* (45 and 68 pairs, respectively). We experimented with other settings, but the results were very similar.

RML's only parameter, $\lambda$, was selected with 5-fold cross-validation. MLKNN's two parameters $k$ (number of neighbors) and $s$ (strength of the uniform prior) were kept fixed to 10 and 1.0, respectively, as was done in [20]. RAkEL's $m$ (number of models) and $t$ (threshold) were set to the library's default (respectively $2 * N$ and 0.5), and $k$ (size of the labelset) was set to $\frac{V}{2}$ as suggested by [4]. For BR we kept the library's defaults.

**Implementation.** Our implementation is in C++, based on the source code of RML[11], which uses the *Bundle Methods for Risk Minimization* (BMRM) of [18]. The max-flow computations needed for graph-cuts are done with the library of [21]. The modifications necessary to enforce positivity in $\theta^2$ in BMRM are described in Appendix C. Source code is available[4] under the Mozilla Public License. Details of training time for our implementation are available in Appendix B.

**Results: *F*-Score.** In Figure 1 we plot the *F*-Score for varying-sized subsets of the unary features, for both *enron* (left) and *yeast* (right). The goal is to assess the benefits of explicitly modelling the pairwise label interactions, particularly when the unary information is deteriorated. As can be seen in Figure 1, when all features are available our model behaves similarly to RML. In this setting the unary features are very informative and the pairwise interactions are not helpful. As we reduce the number of available unary features (from right to left in the plots), the importance of the pairwise interactions increase, and our model demonstrates improvement over RML.

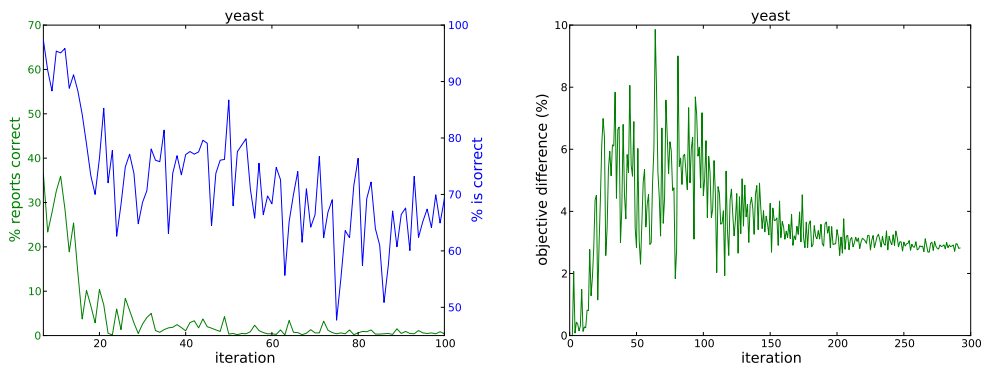

Figure 2: Empirical analysis of Algorithms 2 and 3 during training with the *yeast* dataset. Left: frequency with which Algorithm 2 is optimal at each iteration (blue) and frequency with which Algorithm 3 reports an optimal solution has been found by Algorithm 2 (green). Right: difference, at each iteration, between the objective computed using the results from Algorithm 2 and exhaustive enumeration.

**Results: Correctness.** To evaluate how well our constraint generation algorithm performs in practice we compared its results against those of exhaustive search, which is exact but only feasible for a dataset with a small number of labels, such as *yeast*. We also assessed the strength of our test proposed in Algorithm 3. In Figure 2-left we plot, for the first 100 iterations of the learning algorithm, the frequency with which Algorithm 2 returns the exact solution (blue line) as well as the frequency with which the test given in Algorithm 3 guarantees the solution is exact (green line). We can see that overall in more than 50% of its executions Algorithm 2 produces an optimal solution. Our test effectively offers a lower bound which is as expected is not tight, however it is informative in the sense that its variations reflect legitimate variations in the real quantity of interest (as can be seen by the obvious correlation between the two curves).

For the learning algorithm, however, what we are interested in is the objective $o_i$ and the gradient $g_i$ of line 7 of Algorithm 1, and both depend only on the compound result of $N$ executions of Algorithm 2 at each iteration of the learning algorithm. This is illustrated in Figure 2-right, where we plot, for each iteration, the normalised difference between the objective computed with results from Algorithm 2 and the one computed with the results of an exact exhaustive search[5]. We can see that the difference is quite small – below 4% after the initial iterations.

## 6    Conclusion

We presented a method for learning multi-label classifiers which explicitly models label dependencies in a submodular fashion. As an estimator we use structured support vector machines solved with constraint generation. Our key contribution is an algorithm for constraint generation which is proven to be partially optimal in the sense that all labels it predicts are included in some optimal solution. We also describe an efficiently computable test that if positive guarantees the solution found is optimal, and can be used to generate empirical lower bounds on the probability of finding an optimal solution. We present empirical results that corroborate the fact that the algorithm is very accurate, and we illustrate the gains obtained in comparison to other popular algorithms, particularly a previous algorithm which can be seen as the particular case of ours when there are no explicit label interactions being modelled.

### Acknowledgements

We thank Choon Hui Teo for his help on making the necessary modifications to BMRM. NICTA is funded by the Australian Government as represented by the Department of Broadband, Communications and the Digital Economy and the Australian Research Council through the ICT Centre of Excellence program.

## Footnotes

[1] The worst-case bound of $O(V^3)$ for graph-cuts is very pessimistic; in practice the algorithm is extremely efficient.

[2]http://mulan.sourceforge.net/datasets.html

[3]RML deals mainly with the reverse problem of predicting instances given labels, however it can be applied in the forward direction as well as described in [11].

[4]http://users.cecs.anu.edu.au/~jpetterson/.

[5]We repeated this experiment with several sets of parameters with similar results.

# References

[1] K. Dembczynski, W. Cheng, and E. Hüllermeier, "Bayes Optimal Multilabel Classification via Probabilistic Classifier Chains," in *ICML*, 2010.

[2] X. Zhang, T. Graepel, and R. Herbrich, "Bayesian Online Learning for Multi-label and Multi-variate Performance Measures," in *AISTATS*, 2010.

[3] P. Rai and H. Daume, "Multi-Label Prediction via Sparse Infinite CCA," in *NIPS*, 2009.

[4] J. Read, B. Pfahringer, G. Holmes, and E. Frank, "Classifier chains for multi-label classification.," in *ECML/PKDD*, 2009.

[5] J. Rousu, C. Saunders, S. Szedmak, and J. Shawe-Taylor, "Kernel-based learning of hierarchical multilabel classification models," *JMLR*, vol. 7, pp. 1601–1626, December 2006.

[6] Z. Barutcuoglu, R. E. Schapire, and O. G. Troyanskaya, "Hierarchical multi-label prediction of gene function," *Bioinformatics*, vol. 22, pp. 830–836, April 2006.

[7] M. Guillaumin, T. Mensink, J. Verbeek, and C. Schmid, "TagProp: Discriminative Metric Learning in Nearest Neighbor Models for Image Auto-Annotation," in *ICCV*, 2009.

[8] M. Jansche, "Maximum expected F-measure training of logistic regression models," *HLT*, 2005.

[9] T. Mensink, J. Verbeek, and G. Csurka, "Learning structured prediction models for interactive image labeling," in *CVPR*, 2011.

[10] I. Tsochantaridis, T. Joachims, T. Hofmann, and Y. Altun, "Large margin methods for structured and interdependent output variables," *JMLR*, vol. 6, pp. 1453–1484, 2005.

[11] J. Petterson and T. Caetano, "Reverse multi-label learning," in *NIPS*, 2010.

[12] W. Bi and J. Kwok, "Multi-Label Classification on Tree- and DAG-Structured Hierarchies," in *ICML*, 2011.

[13] N. Ghamrawi and A. Mccallum, "Collective Multi-Label Classification," 2005.

[14] B. Hariharan, S. V. N. Vishwanathan, and M. Varma, "Large Scale Max-Margin Multi-Label Classification with Prior Knowledge about Densely Correlated Labels," in *ICML*, 2010.

[15] G. Tsoumakas, I. Katakis, and I. P. Vlahavas, *Mining Multi-label Data*. Springer, 2009.

[16] G. Tsoumakas and I. P. Vlahavas, "Random k-labelsets: An ensemble method for multilabel classification," in *ECML*, 2007.

[17] S. Virtanen, A. Klami, and S. Kaski, "Bayesian CCA via Group Sparsity," in *ICML*, 2011.

[18] C. H. Teo, S. V. N. Vishwanathan, A. J. Smola, and Q. V. Le, "Bundle methods for regularized risk minimization," *JMLR*, vol. 11, pp. 311–365, 2010.

[19] Z. Svitkina and L. Fleischer, "Submodular approximation: Sampling-based algorithms and lower bounds," in *FOCS*, 2008.

[20] M.-L. Zhang and Z.-H. Zhou, "ML-KNN: A lazy learning approach to multi-label learning," *Pattern Recognition*, vol. 40, pp. 2038–2048, July 2007.

[21] Y. Boykov and V. Kolmogorov, "An experimental comparison of min-cut/max-flow algorithms for energy minimization in vision," *IEEE Trans. PAMI*, 2004.

